# Recognizing Overlapping Hand-Printed Characters by Centered-Object Integrated Segmentation and Recognition

**Gale L. Martin\* & Mosfeq Rashid**
MCC
Austin, Texas 78759 USA

## Abstract

This paper describes an approach, called *centered object integrated segmentation and recognition* (COISR), for integrating object segmentation and recognition within a single neural network. The application is hand-printed character recognition. Two versions of the system are described. One uses a backpropagation network that scans exhaustively over a field of characters and is trained to recognize whether it is centered over a single character or between characters. When it is centered over a character, the net classifies the character. The approach is tested on a dataset of hand-printed digits. Very low error rates are reported. The second version, COISR-SACCADE, avoids the need for exhaustive scans. The net is trained as before, but also is trained to compute ballistic 'eye' movements that enable the input window to jump from one character to the next.

The common model of visual processing includes multiple, independent stages. First, filtering operations act on the raw image to segment or isolate and enhance to-be-recognized clumps. These clumps are normalized for factors such as size, and sometimes simplified further through feature extraction. The results are then fed to one or more classifiers. The operations prior to classification simplify the recognition task. Object segmentation restricts the number of features considered for classification to those associated with a single object, and enables normalization to be applied at the individual object level. Without such pre-processing, recognition may be an intractable problem. However, a weak point of this sequential stage model is that recognition and segmentation decisions are often inter-dependent. Not only does a correct recognition decision depend on first making a correct segmentation decision, but a correct segmentation decision often depends on first making a correct recognition decision.

This is a particularly serious problem in character recognition applications. OCR systems use intervening white space and related features to segment a field of characters into individual characters, so that classification can be accomplished one character at a time. This approach fails when characters touch each other or when an individual character is broken up by intervening white space. Some means of integrating the segmentation and recognition stages is needed.

This paper describes an approach, called *centered object integrated segmentation and recognition* (COISR), for integrating character segmentation and recognition within one

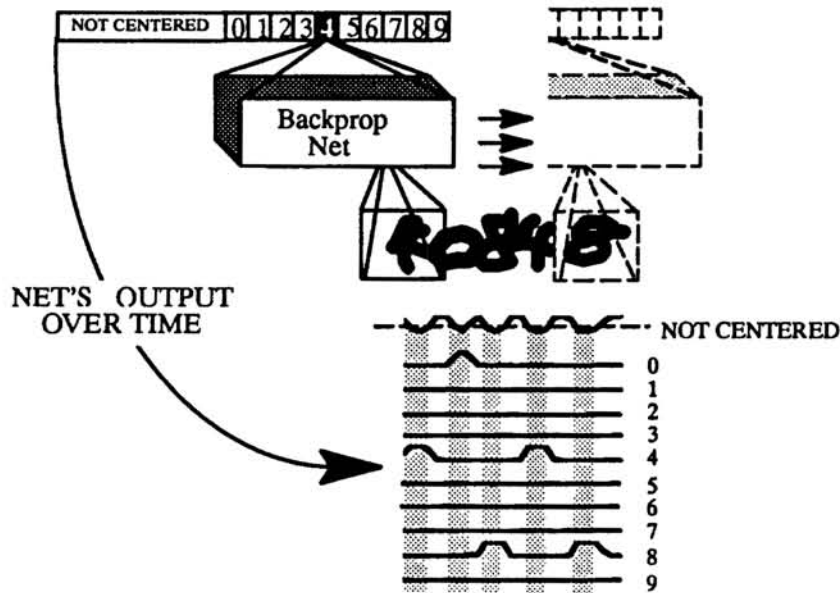

Figure 1: The COISR Exhaustive Scan Approach.

neural network.   The general approach builds on previous work in pre-segmented character recognition (LeCun, Boser, Denker, Henderson, Howard, Hubbard, & Jackel, 1990; Martin & Pittman, 1990) and on the sliding window conception used in neural network speech applications, such as NETtalk (Sejnowski & Rosenberg(1986) and Time Delay Neural Networks (Waibel, Sawai, & Shikano, 1988). Two versions of the approach are described.  In both cases, a net is trained to recognize what is centered in its input window as it slides along a character field.  The window size is chosen to be large enough to include more than one character.

# 1   COISR VERSION 1:  EXHAUSTIVE SCAN

As shown in Figure 1, the net is trained on an input window, and a target output vector representing what is in the center of the window. The top half of the figure shows the net's input window scanning successively across the field. Sometimes the middle of the window is centered on a character, and sometimes it is centered on a point between two characters. The target output vector consists of one node per category, and one node corresponding to a NOT-CENTERED condition.  This latter node has a high target activation value when the input window is not centered over any character.  A temporal stream of output vectors is created (shown at the bottom half of the figure) as the net scans the field. There is no need to explicitly segment characters, during training or testing, because recognition is defined as identifying what is in the center of the scanning window.  The net learns to extract regularities in the shapes of individual characters even when those regularities occur in the context of overlapping and broken characters. The final stage of processing involves parsing the temporal stream generated as the net scans the field to yield an ascii string of recognized characters.

## 1.1 IMPLEMENTATION DETAILS

The COISR approach was tested using the National Institute of Standards and Technology (NIST) database of hand-printed digit fields,  using fields 6–30 of the form, which correspond to five different fields of length 2, 3, 4, 5, or 6 digits each.  The training data included roughly 80,000 digits (800 forms, 20,000 fields), and came from forms labeled f0000–f0499, and f1500–f1799 in the dataset.  The test data consisted of roughly 20,000 digits (200 forms, 5,000 fields) and came from forms labeled f1800–f1899 and f2000–f2099 in the dataset.  The large test set was used because considerable variations

in test scores occurred with smaller test set sizes. The samples were scanned at a 300 pixel/inch resolution. Each field image was preprocessed to eliminate the white space and box surrounding the digit field. Each field was then size normalized with respect to the vertical height of the digit field to a vertical height of 20 pixels. Since the input is size normalized to the vertical height of the field of characters, the actual number of characters in the constant-width input window of 36 pixels varies depending on the height-to-width ratio for each character. The scan rate was a 3-pixel increment across the field.

A key design principle of the present approach is that highly accurate integrated segmentation and recognition requires training on both the shapes of characters and their positions within the input window. The field images used for training were labeled with the horizontal center positions of each character in the field. The human labeler simply pointed at the horizontal center of each digit in sequence with a mouse cursor and clicked on a mouse button. The horizontal position of each character was then paired with its category label (0-9) in a data file. The labeling process is not unlike a human reading teacher using a pointer to indicate the position of each character as he or she reads aloud the sequence of characters making up the word or sentence. During testing this position information is not used.

Position information about character centers is used to generate target output values for each possible position of the input window as it scans a field of characters. When the center position of a window is close to the center of a character, the target value of that character's output node is set at the maximum, with the target value of the NOT-CENTERED node set at the minimum. The activation values of all other characters' output nodes are set at the minimum. When the center position of a window is close to the half-way point between two character centers, the target value of all character output nodes are set to the minimum and the target value of the NOT-CENTERED node is set to a maximum. Between these two extremes, the target values vary linearly with distance, creating a trapezoidal function (i.e., ).

The neural network is a 2-hidden-layer backpropagation network, with local, shared connections in the first hidden layer, and local connections in the second hidden layer (see Figure 2). The first hidden layer consists of 2016 nodes, or more specifically 18 independent groups of 112 (16x7) nodes, with each group having local, shared connec-

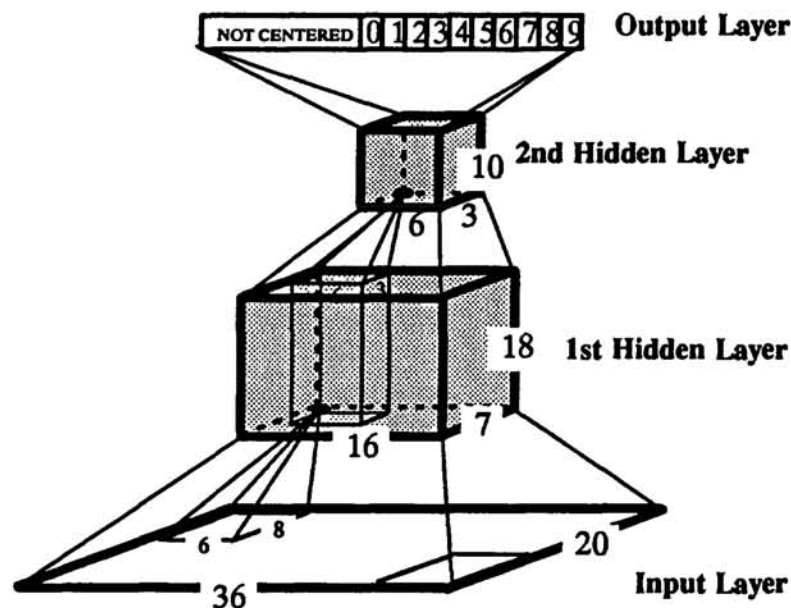

Figure 2: Architecture for the COISR-Exhaustive Scan Approach.

tions to the input layer. The local, overlapping receptive fields of size 6x8 are offset by 2 pixels, such that the region covered by each group of nodes spans the input layer. The second hidden layer consists of 180 nodes, having local, but NOT shared receptive fields of size 6x3. The output layer consists of 11 nodes, with each of these nodes connected to all of the nodes in the 2nd hidden layer. The net has a total of 2927 nodes (includes input and output nodes), and 157,068 connections. In a feedforward (non-learning) mode on a DEC 5000 workstation, in which the net is scanning a field of digits, the system processes about two digits per second, which includes image pre-processing and the necessary number of feedforward passes on the net.

As the net scans horizontally, the activation values of the 11 output nodes create a trace as shown in Figure 1. To convert this to an ascii string corresponding to the digits in the field, the state of the NOT–CENTERED node is monitored continuously. When it's activation value falls below a threshold, a summing process begins for each of the other nodes, and ends when the activation value of the NOT–CENTERED node exceeds the threshold. At this point the system decides that the input window has moved off of a character. The system then classifies the character on the basis of which output node has the highest summed activation for the position just passed over.

## 1.2 GENERALIZATION PERFORMANCE

As shown in Figure 3, the COISR technique achieves very low field–based error rates,

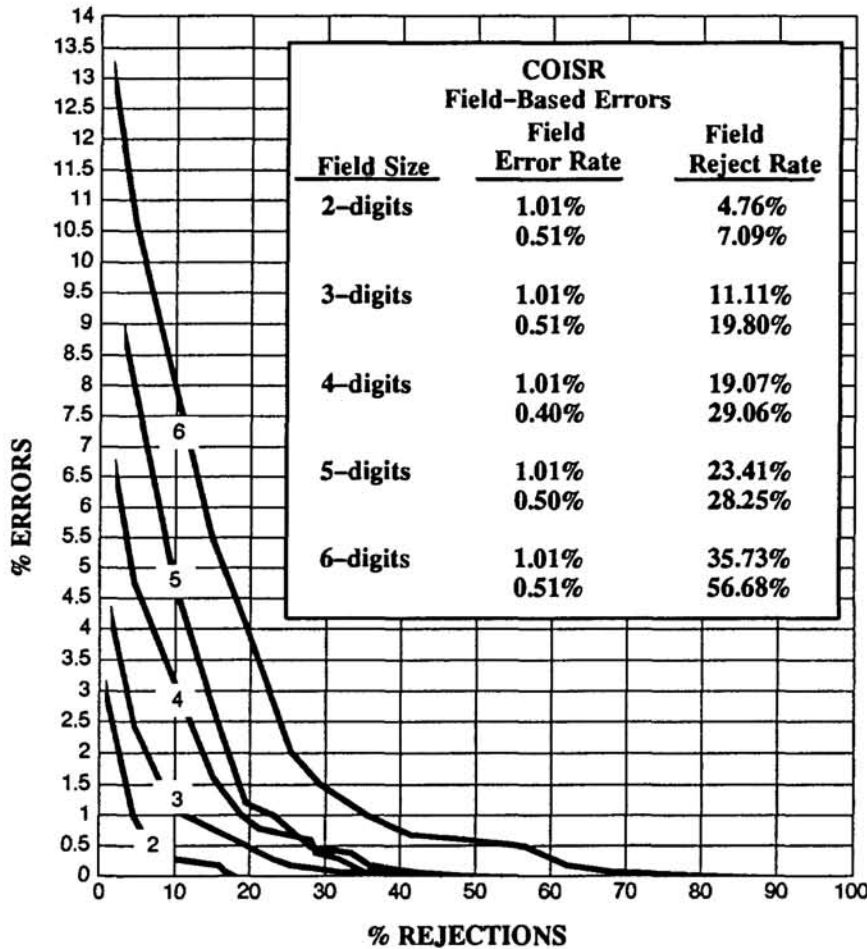

| | COISR Field-Based Errors | |
|---|---|---|
| Field Size | Field Error Rate | Field Reject Rate |
| 2–digits | 1.01% | 4.76% |
| | 0.51% | 7.09% |
| 3–digits | 1.01% | 11.11% |
| | 0.51% | 19.80% |
| 4–digits | 1.01% | 19.07% |
| | 0.40% | 29.06% |
| 5–digits | 1.01% | 23.41% |
| | 0.50% | 28.25% |
| 6–digits | 1.01% | 35.73% |
| | 0.51% | 56.68% |

Figure 3: Field–based Test Error and Reject Rates

particularly for a single classifier system. The error rates are field–based in the sense that if the network mis–classifies one character in the field, the entire field is consid-

ered as mis-classified. Error rates pertain to the fields remaining, after rejection. Rejections are based on placing a threshold for the acceptable distance between the highest and the next highest running activation total. In this way, by varying the threshold, the error rate can be traded off against the percentage of rejections. Since the reported data apply to fields, the threshold applies to the smallest distance value found across all of the characters in the field. Figure 4 provides examples, from the test set, of fields that the COISR network correctly classifies.

Figure 4: Test Set Examples of Touching and Broken Characters Correctly Recognized

The COISR technique is a success in the sense that it does something that conventional character recognition systems can not do. It robustly recognizes character fields containing touching, overlapping, and broken characters. One problem with the approach, however, lies with the exhaustive nature of the scan. The components needed to recognize a character in a given location are essentially replicated across the length of the to-be-classified input field, at the degree of resolution necessary to recognize the smallest and closest characters. While this has not presented any real difficulties for the present system, which processes 2 characters per second, it is likely to be troublesome when extensions are made to two-dimensional scans and larger vocabularies. A rough analogy with respect to human vision would be to require that all of the computational resources needed for recognizing objects at one point on the retina be replicated for each resolvable point on the retina. This design carries the notion of a compound eye to the ridiculous extreme.

## 2   COISR VERSION 2: SACCADIC SCAN

Taking a cue from natural vision systems, the second version of the COISR system uses a *saccadic scan*. The system is trained to make ballistic eye movements, so that it can effectively jump from character to character and over blank areas. This version is similar to the exhaustive scan version in the sense that a backprop net is trained to recognize when it's input window is centered on a character, and if so, to classify the character. In addition, the net is trained for navigation control (Pomerleau, 1991). At each point in a field of characters, the net is trained to estimate the distance to the next character on the right, and to estimate the degree to which the center-most character is off-center. The trained net accomodates for variations in character width, spacing between characters, writing styles, and other factors. At run-time, the system uses the computed character classification and distances to navigate along a character field. If the character classification judgment, for a given position, has a high degree of certainty, the system accesses the *next character* distance information computed by the net for the current position and executes the jump.    If the system gets off-track, so that a

character can not be recognized with a high–degree of certainty, it makes a corrective saccade by accessing the *off–center character* distance computed by the net for the current position. This action corresponds to making a second attempt to center the character within the input window.

The primary advantage of this approach, over the exhaustive scan, is improved efficiency, as illustrated in Figure 5. The scanning input windows are shown at the top of the figure, for each approach, and each character–containing input window, shown below the scanned image for each approach, corresponds to a forward pass of the net. The exhaustive scan version requires about 4 times as many forward passes as the saccadic scan version. Greater improvements in efficiency can be achieved with wider input windows and images containing more blank areas. The system is still under development, but accuracy approaches that of the exhaustive scan system.

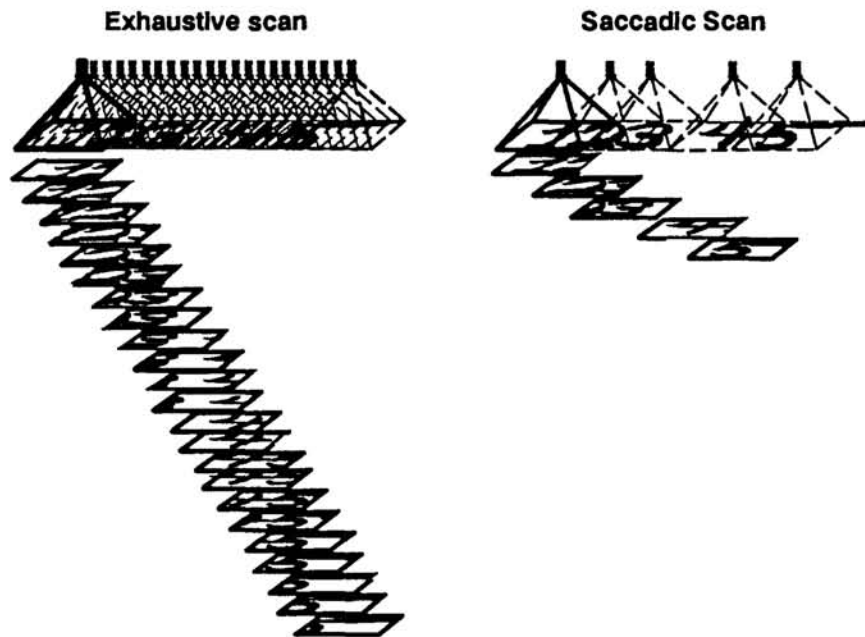

Figure 5: Number of Forward Passes for Saccadic & Exhaustive Scan Systems

## 3  COMPARISONS & CONCLUSIONS

In comparing accuracy rates between different OCR systems, one relevant factor that should be reported is the number of classifiers used. For a given system, increasing the number of classifiers typically reduces error rates but increases processing time. The low error rates reported here for the COISR-Exhaustive Scan approach come from a single classifier operating at 2 characters per second on a general purpose workstation. Most OCR systems employ multiple classifiers. For example, at the NIPS workshops this year, Jonathan Hull described the University of Buffalo zip code recognition system that contains five classifiers and requires about one minute to process a character. Keeler and Rumelhart, at this conference, also described a two–classifier neural net system for NIST digit recognition. The fact that the COISR approach achieved quite low error rates with a single classifier indicates that the approach is a promising one.

Clearly, another relevant factor in comparing systems is the ability to recognize touching and broken characters, since this is a dominant stumbling block for current OCR systems. Conventional systems can be altered to achieve integrated segmentation and recognition in limited cases, but this involves a lot of hand–crafting and a significant

amount of time–consuming iterative processing (Fenrich. 1991). Essentially, multiple segmenters are used, and classification is performed for each such possible segmentation. The final segmentation and recognition decisions can thus be inter-dependent, but only at the cost of computing multiple segmentations and correspondingly, multiple classification decisions. The approach breaks down as the number of possible segmentations increases, as would occur for example if individual characters are broken or touching in multiple places or if multiple letters in a sequence are connected. The COISR system does not appear to have this problem.

The NIPS conference this year has included a number of other neural net approaches to integrated segmentation and recognition in OCR domains. Two approaches similar to the COISR–Exhaustive Scan system were those described by Faggin and by Keeler and Rumelhart. All three achieve integrated segmentation and recognition by convolving a neural network over a field of characters. Faggin described an analog hardware implementation of a neural-network-based OCR system that receives as input a window that slides along the machine–print digit field at the bottom of bank checks. Keeler and Rumelhart described a *self–organizing integrated segmentation and recognition* (SOISR) system. Initially, it is trained on characters that have been pre-segmented by a labeler effectively drawing a box around each. Then, in subsequent training, a net, with these pre-trained weights, is duplicated repetitively across the extent of a fixed–width input field, and is further trained on examples of entire fields that contain connecting or broken characters.

All three approaches have the weakness, described previously of performing essentially exhaustive scans or convolutions over the to–be–classified input field. This complaint is not necessarily directed at the specific applications dealt with at this year's NIPS conference, particularly if operating at the high levels of efficiency described by Faggin. Nor is the complaint directed at tasks that only require the visual system to focus on a few small clusters or fields in the larger, otherwise blank input field. In these cases, low–resolution filters may be sufficient to efficiently remove blank areas and enable efficient integrated segmentation and recognition. However, we use as an example, the saccadic scanning behavior of human vision in tasks, such as reading this paragraph. In such cases that require high–resolution sensitivity across a large, dense image and classification of a very large vocabulary of symbols, it seems clear that other, more flexible and efficient scanning mechanisms will be necessary. This high–density image domain is the focus of the COISR–Saccadic Scan approach, which integrates not only the segmentation and recognition of characters, but also control of the navigational aspects of vision.

## Acknowledgements

We thank Lori Barski, John Canfield, David Chapman, Roger Gaborski, Jay Pittman, and Dave Rumelhart for helpful discussions and/or development of supporting image handling and network software. I also thank Jonathan Martin for help with the position labeling.

## Footnotes

\*Also with Eastman Kodak Company

## References

Fenrich, R. Segmentation of automatically located handwritten words. paper presented at the *International Workshop on frontiers in handwriting recognition*. Chateau de Bonas, France. 23–27 September 1991.

Keeler, J. D., Rumelhart, David E., Leow, Wee–Kheng. Integrated segmentation and recognition of hand–printed numerals, in R. P. Lippmann, John E. Moody, David S. Touretzky (Eds) *Advances in Neural Information Processing Systems 3*, p.557–563. San Mateo, CA: Morgan Kaufmann. 1991.

LeCun, Y., Boser, B., Denker, J., Henderson, D., Howard, R. E., Hubbard, W. & Jackel, L. D. Handwritten digit recognition with a backpropagation network, in D. S. Touretzky (Ed.) *Advances in Neural Information Processing Systems 2*. Morgan Kaufmann, 1990.

Martin, G. L. & Pittman, J. A. Recognizing hand-printed letters and digits. in D. S. Touretzky (Ed.) *Advances in Neural Information Processing Systems 2*. Morgan Kaufmann, 1990.

Pomerleau, D. A. Efficient training of artificial neural networks for autonomous navigation. *Neural Computation, 3*, 1991, 88-97.

Rumelhart, D. (1989) Learning and generalization in multi-layer networks. presentation given at the NATO Advanced Research Workshop on Neuro Computing Algorithms, Architectures and Applications. Les Arcs, France. February, 1989.

Sejnowski, T. J. & Rosenberg, C. R. (1986) NETtalk: a parallel network that learns to read aloud. Johns Hopkins University Electrical Engineering and Computer Science Technical Report JHU/EECS-86/01.

Waibel, A., Sawai, H., Shikano, K. (1988) Modularity and scaling in large phonemic neural networks. ATR Interpreting Telephony Research Laboratories Technical Report TR-I-0034.